# Symmetric Correspondence Topic Models for Multilingual Text Analysis

**Kosuke Fukumasu**[†]     **Koji Eguchi**[†]     **Eric P. Xing**[‡]

[†]Graduate School of System Informatics, Kobe University, Kobe 657-8501, Japan
[‡]School of Computer Science, Carnegie Mellon University, Pittsburgh, PA 15213, USA
`fukumasu@cs25.scitec.kobe-u.ac.jp, eguchi@port.kobe-u.ac.jp, epxing@cs.cmu.edu`

## Abstract

Topic modeling is a widely used approach to analyzing large text collections. A small number of multilingual topic models have recently been explored to discover latent topics among parallel or comparable documents, such as in Wikipedia. Other topic models that were originally proposed for structured data are also applicable to multilingual documents. Correspondence Latent Dirichlet Allocation (CorrLDA) is one such model; however, it requires a pivot language to be specified in advance. We propose a new topic model, Symmetric Correspondence LDA (SymCorrLDA), that incorporates a hidden variable to control a pivot language, in an extension of CorrLDA. We experimented with two multilingual comparable datasets extracted from Wikipedia and demonstrate that SymCorrLDA is more effective than some other existing multilingual topic models.

## 1 Introduction

Topic models (also known as mixed-membership models) are a useful method for analyzing large text collections [1, 2]. In topic modeling, each document is represented as a mixture of topics, where each topic is represented as a word distribution. Latent Dirichlet Allocation (LDA) [2] is one of the well-known topic models. Most topic models assume that texts are monolingual; however, some can capture statistical dependencies between multiple classes of representations and can be used for multilingual parallel or comparable documents. Here, a *parallel document* is a merged document consisting of multiple language parts that are translations from one language to another, sometimes including sentence-to-sentence or word-to-word alignments. A *comparable document* is a merged document consisting of multiple language parts that are not translations of each other but instead describe similar concepts and events. Recently published multilingual topic models [3, 4], which are the equivalent of Conditionally Independent LDA (CI-LDA) [5, 6], can discover latent topics among parallel or comparable documents. SwitchLDA [6] was modeled by extending CI-LDA. It can control the proportions of languages in each multilingual topic. However, both CI-LDA and SwitchLDA preserve dependencies between languages only by sharing per-document multinomial distributions over latent topics, and accordingly the resulting dependencies are relatively weak.

Correspondence LDA (CorrLDA) [7] is another type of topic model for structured data represented in multiple classes. It was originally proposed for annotated image data to simultaneously model words and visual features, and it can also be applied to parallel or comparable documents. In the modeling, it first generates topics for visual features in an annotated image. Then only the topics associated with the visual features in the image are used to generate words. In this sense, visual features can be said to be the *pivot* in modeling annotated image data. However, when CorrLDA is applied to multilingual documents, a language that plays the role of the pivot (a pivot language[1])

must be specified in advance. The pivot language selected is sensitive to the quality of the multi-lingual topics estimated with CorrLDA. For example, a translation of a Japanese book into English would presumably have a pivot to the Japanese book, but a set of international news stories would have pivots that differ based on the country an article is about. It is often difficult to appropriately select the pivot language. To address this problem, which we call the *pivot problem*, we propose a new topic model, Symmetric Correspondence LDA (SymCorrLDA), that incorporates a hidden variable to control the pivot language, in an extension of CorrLDA. Our SymCorrLDA addresses the problem of CorrLDA and can select an appropriate pivot language by inference from the data.

We evaluate various multilingual topic models, i.e., CI-LDA, SwitchLDA, CorrLDA, and our Sym-CorrLDA, as well as LDA, using comparable articles in different languages (English, Japanese, and Spanish) extracted from Wikipedia. We first demonstrate through experiments that CorrLDA outper-forms the other existing multilingual topic models mentioned, and then show that our SymCorrLDA works more effectively than CorrLDA in any case of selecting a pivot language.

## 2   Multilingual Topic Models with Multilingual Comparable Documents

Bilingual topic models for bilingual parallel documents that have word-to-word alignments have been developed, such as those by [8]. Their models are directed towards machine translation, where word-to-word alignments are involved in the generative process. In contrast, we focus on analyzing dependencies among languages by modeling multilingual comparable documents, each of which consists of multiple language parts that are not translations of each other but instead describe similar concepts and events. The target documents can be parallel documents, but word-to-word alignments are not taken into account in the topic modeling. Some other researchers explored different types of multilingual topic models that are based on the premise of using multilingual dictionaries or WordNet [9, 10, 11]. In contrast, CI-LDA and SwitchLDA only require multilingual comparable documents that can be easily obtained, such as from Wikipedia, when we use those models for multilingual text analysis. This is more similar to the motivation of this paper. Below, we introduce LDA-style topic models that handle multiple classes and can be applied to multilingual comparable documents for the above-mentioned purposes.

### 2.1   Conditionally Independent LDA (CI-LDA)

CI-LDA [5, 6] is an extension of the LDA model to handle multiple classes, such as words and citations in scientific articles. The CI-LDA framework was used to model multilingual parallel or comparable documents by [3] and [4]. Figure 1(b) shows a graphical model representation of CI-LDA for documents in $L$ languages, and Figure 1(a) shows that of LDA for reference. $D$, $T$, and $N_d$ respectively indicate the number of documents, number of topics, and number of word tokens that appear in a specific language part in a document $d$. The superscript '$(\cdot)$' indicates the variables corresponding to a specific language part in a document $d$. For better understanding, we show below the process of generating a document according to the graphical model of the CI-LDA model.

1. For all $D$ documents, sample $\boldsymbol{\theta}_d \sim Dirichlet(\boldsymbol{\alpha})$
2. For all $T$ topics and for all $L$ languages, sample $\phi_t^{(\ell)} \sim Dirichlet(\boldsymbol{\beta}^{(\ell)})$
3. For each of the $N_d^{(\ell)}$ words $w_i^{(\ell)}$ in language $\ell$ ($\ell \in \{1, \cdots, L\}$) of document $d$:
   a. Sample a topic $z_i^{(\ell)} \sim Multinomial(\boldsymbol{\theta}_d)$
   b. Sample a word $w_i^{(\ell)} \sim Multinomial(\phi_{z_i^{(\ell)}}^{(\ell)})$

For example, when we deal with Japanese and English bilingual data, $w^{(1)}$ and $w^{(2)}$ are a Japanese and an English word, respectively. CI-LDA preserves dependencies between languages only by sharing the multinomial distributions with parameters $\boldsymbol{\theta}_d$. Accordingly, there are substantial chances that some topics are assigned only to a specific language part in each document, and the resulting dependencies are relatively weak.

### 2.2   SwitchLDA

Similarly to CI-LDA, SwitchLDA [6] can be applied to multilingual comparable documents. How-ever, different from CI-LDA, SwitchLDA can adjust the proportions of multiple different languages for each topic, according to a binomial distribution for bilingual data or a multinomial distribu-tion for data of more than three languages. Figure 1(c) depicts a graphical model representation of SwitchLDA for documents in $L$ languages. The generative process is described below.

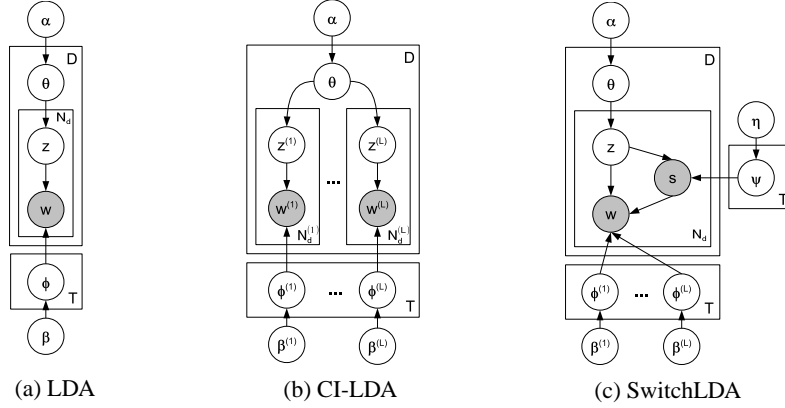

<div align="center">
(a) LDA      (b) CI-LDA      (c) SwitchLDA

Figure 1: Graphical model representations of (a) LDA, (b) CI-LDA, and (c) SwitchLDA
</div>

1. For all $D$ documents, sample $\boldsymbol{\theta}_d \sim Dirichlet(\boldsymbol{\alpha})$
2. For all $T$ topics:
   a. For all $L$ languages, sample $\phi_t^{(\ell)} \sim Dirichlet(\boldsymbol{\beta}^{(\ell)})$
   b. Sample $\boldsymbol{\psi}_t \sim Dirichlet(\boldsymbol{\eta})$
3. For each of the $N_d$ words $w_i$ in document $d$:
   a. Sample a topic $z_i \sim Multinomial(\boldsymbol{\theta}_d)$
   b. Sample a language label $s_i \sim Multinomial(\boldsymbol{\psi}_{z_i})$
   c. Sample a word $w_i \sim Multinomial(\phi_{z_i}^{(s_i)})$

Here, $\boldsymbol{\psi}_t$ indicates a multinomial parameter to adjust the proportions of $L$ different languages for topic $t$. If all components of hyperparameter vector $\boldsymbol{\eta}$ are large enough, SwitchLDA becomes equivalent to CI-LDA. SwitchLDA is an extension of CI-LDA to give emphasis or de-emphasis to specific languages for each topic. Therefore, SwitchLDA may represent multilingual topics more flexibly; however, it still has the drawback that the dependencies between languages are relatively weak.

## 2.3 Correspondence LDA (CorrLDA)

CorrLDA [7] can also be applied to multilingual comparable documents. In the multilingual setting, this model first generates topics for one language part of a document. We refer to this language as a *pivot language*. For the other languages, the model then uses the topics that were already generated in the pivot language. Figure 2(a) shows a graphical model representation of CorrLDA assuming $L$ languages, when $p$ is the pivot language that is specified in advance. Here, $N_d^{(\ell)}$ ($\ell \in \{p, 2, \cdots, L\}$) denotes the number of words in language $\ell$ in document $d$. The generative process is shown below:

1. For all $D$ documents' pivot language parts, sample $\boldsymbol{\theta}_d^{(p)} \sim Dirichlet(\boldsymbol{\alpha}^{(p)})$
2. For all $T$ topics and for all $L$ languages (including the pivot language), sample $\phi_t^{(\ell)} \sim Dirichlet(\boldsymbol{\beta}^{(\ell)})$
3. For each of the $N_d^{(p)}$ words $w_i^{(p)}$ in the pivot language $p$ of document $d$:
   a. Sample a topic $z_i^{(p)} \sim Multinomial(\boldsymbol{\theta}_d^{(p)})$
   b. Sample a word $w_i^{(p)} \sim Multinomial(\phi_{z_i^{(p)}}^{(p)})$
4. For each of the $N_d^{(\ell)}$ words $w_i^{(\ell)}$ in language $\ell$ ($\ell \in \{2, \cdots, L\}$) of document $d$:
   a. Sample a topic $y_i^{(\ell)} \sim Uniform\left(z_1^{(p)}, \cdots, z_{N_d^{(p)}}^{(p)}\right)$
   b. Sample a word $w_i^{(\ell)} \sim Multinomial(\phi_{y_i^{(\ell)}}^{(\ell)})$

This model can capture more direct dependencies between languages, due to the constraints that topics have to be selected from the topics selected in the pivot language parts. However, when CorrLDA is applied to multilingual documents, a pivot language must be specified in advance. Moreover, the pivot language selected is sensitive to the quality of the multilingual topics estimated with CorrLDA.

## 3 Symmetric Correspondence Topic Models

When CorrLDA is applied to parallel or comparable documents, this model first generates topics for one language part of a document, which we refer to this language as a *pivot language*. For the other languages, the model then uses the topics that were already generated in the pivot language. CorrLDA has the great advantage that it can capture more direct dependency between languages;

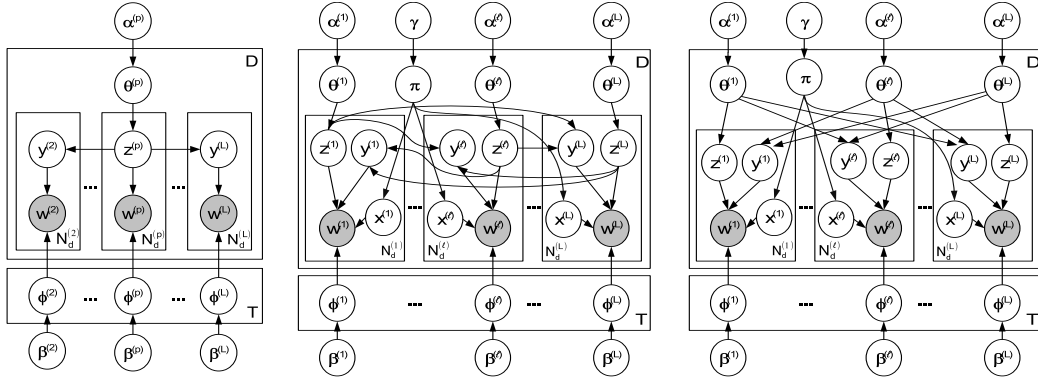

(a) CorrLDA        (b) SymCorrLDA        (c) alternative SymCorrLDA

Figure 2: Graphical model representations of (a) CorrLDA, (b) SymCorrLDA, and (c) its variant

however, it has a disadvantage that it requires a pivot language to be specified in advance. Since the pivot language may differ based on the subject, such as the country a document is about, it is often difficult to appropriately select the pivot language. To address this problem, we propose Symmetric Correspondence LDA (SymCorrLDA). This model generates a flag that specifies a pivot language for each word, adjusting the probability of being pivot languages in each language part of a document according to a binomial distribution for bilingual data or a multinomial distribution for data of more than three languages. In other words, SymCorrLDA estimates from the data the best pivot language at the word level in each document. The pivot language flags may be assigned to the words in the originally written portions in each language, since the original portions may be described confidently and with rich vocabulary. Figure 2(b) shows a graphical model representation of SymCorrLDA. SymCorrLDA's generative process is shown as follows, assuming $L$ languages:

1. For all $D$ documents:
   a. For all $L$ languages, sample $\boldsymbol{\theta}_d^{(\ell)} \sim Dirichlet(\boldsymbol{\alpha}^{(\ell)})$
   b. Sample $\boldsymbol{\pi}_d \sim Dirichlet(\boldsymbol{\gamma})$
2. For all $T$ topics and for all $L$ languages, sample $\boldsymbol{\phi}_t^{(\ell)} \sim Dirichlet(\boldsymbol{\beta}^{(\ell)})$
3. For each of the $N_d^{(\ell)}$ words $w_i^{(\ell)}$ in language $\ell$ ($\ell \in \{1, \cdots, L\}$) of document $d$:
   a. Sample a pivot language flag $x_i^{(\ell)} \sim Multinomial(\boldsymbol{\pi}_d)$
   b. If $(x_i^{(\ell)}=\ell)$, sample a topic $z_i^{(\ell)} \sim Multinomial(\boldsymbol{\theta}_d^{(\ell)})$
   c. If $(x_i^{(\ell)}=m\neq\ell)$, sample a topic $y_i^{(\ell)} \sim Uniform\left(z_1^{(m)}, \cdots, z_{M_d^{(m)}}^{(m)}\right)$
   d. Sample a word $w_i^{(\ell)} \sim Multinomial\left(\delta_{x_i^{(\ell)}=\ell}\phi_{z_i^{(\ell)}}^{(\ell)} + (1 - \delta_{x_i^{(\ell)}=\ell})\phi_{y_i^{(\ell)}}^{(\ell)}\right)$

The pivot language flag $x_i^{(\ell)} = \ell$ for an arbitrary language $\ell$ indicates that the pivot language for the word $w_i^{(\ell)}$ is its own language $\ell$, and $x_i^{(\ell)} = m$ indicates that the pivot language for $w_i^{(\ell)}$ is another language $m$ different from its own language $\ell$. The indicator function $\delta$ takes the value 1 when the designated event occurs and 0 if otherwise. Unlike CorrLDA, the uniform distribution at Step 3-c is not based on the topics that are generated for all $N_d^{(m)}$ words with the pivot language flags, but based only on the topics that are already generated for $M_d^{(m)}(M_d^{(m)} \leq N_d^{(m)})$ words with the pivot language flags at each step while in the generative process.[2] The full conditional probability for collapsed Gibbs sampling of this model is given by the following equations, assuming symmetric Dirichlet priors parameterized by $\alpha^{(\ell)}, \beta^{(\ell)}$ ($\ell \in \{1, \cdots, L\}$), and $\gamma$:

$$P(z_i^{(\ell)} = t, x_i^{(\ell)} = \ell | \mathbf{w}_i^{(\ell)} = w^{(\ell)}, \mathbf{z}_{-i}^{(\ell)}, \mathbf{w}_{-i}^{(\ell)}, \mathbf{x}_{-i}, \alpha^{(\ell)}, \beta^{(\ell)}, \gamma) \propto$$

$$\frac{n_{d\ell,-i} + \gamma}{n_{d\ell,-i} + \sum_{j\neq\ell} n_{dj} + L\gamma} \cdot \frac{C_{td,-i}^{TD^{(\ell)}} + \alpha^{(\ell)}}{\sum_{t'} C_{t'd,-i}^{TD^{(\ell)}} + T\alpha^{(\ell)}} \cdot \frac{C_{w^{(\ell)}t,-i}^{W^{(\ell)}T} + \beta^{(\ell)}}{\sum_{w^{(\ell)}\prime} C_{w^{(\ell)}\prime t,-i}^{W^{(\ell)}T} + W^{(\ell)}\beta^{(\ell)}} \tag{1}$$

$$P(y_i^{(\ell)} = t, x_i^{(\ell)} = m | \mathbf{w}_i^{(\ell)} = w^{(\ell)}, \mathbf{y}_{-i}^{(\ell)}, \mathbf{z}^{(m)}, \mathbf{w}_{-i}^{(\ell)}, \mathbf{x}_{-i}, \beta^{(\ell)}, \gamma) \propto$$

| Table 1: Summary of bilingual data | | |
|---|---|---|
| | Japanese | English |
| No. of documents | 229,855 | |
| No. of word types (vocab) | 124,046 | 173,157 |
| No. of word tokens | 61,187,469 | 80,096,333 |

| Table 2: Summary of trilingual data | | | |
|---|---|---|---|
| | Japanese | English | Spanish |
| No. of documents | 90,602 | | |
| No. of word types (vocab) | 70,902 | 98,474 | 96,191 |
| No. of word tokens | 25,952,978 | 33,999,988 | 25,701,830 |

$$\frac{n_{dm,-i} + \gamma}{n_{dm,-i} + \sum_{j \neq m} n_{dj} + L\gamma} \cdot \frac{C_{td}^{TD^{(m)}}}{N_d^{(m)}} \cdot \frac{C_{w^{(\ell)'} t,-i}^{W^{(\ell)}T} + \beta^{(\ell)}}{\sum_{w^{(\ell)'}} C_{w^{(\ell)'} t,-i}^{W^{(\ell)}T} + W^{(\ell)}\beta^{(\ell)}} \tag{2}$$

where $\mathbf{w}^{(\cdot)} = \{w_i^{(\cdot)}\}$, $\mathbf{z}^{(\cdot)} = \{z_i^{(\cdot)}\}$, and $\mathbf{x}^{(\cdot)} = \{x_i^{(\cdot)}\}$. $W^{(\cdot)}$ and $N_d^{(\cdot)}$ respectively indicate the total number of vocabulary words (word types) in the specified language, and the number of word tokens that appear in the specified language part of document $d$. $n_{d\ell}$ and $n_{dm}$ are the number of times, for an arbitrary word $i \in \{1, \cdots, N_d^{(\cdot)}\}$ in an arbitrary language $j \in \{1, \cdots, L\}$ of document $d$, the flags $x_i^{(j)} = \ell$ and $x_i^{(j)} = m$ respectively are allocated to document $d$. $C_{td}^{TD^{(\cdot)}}$ indicates the $(t, d)$ element of a $T \times D$ topic-document count matrix, meaning the number of times topic $t$ is allocated to the document $d$'s language part specified in parentheses. $C_{wt}^{W^{(\cdot)}T}$ indicates the $(w, t)$ element of a $W^{(\cdot)} \times T$ word-topic count matrix, meaning the number of times topic $t$ is allocated to word $w$ in the language specified in parentheses. The subscript '$-i$' indicates when $w_i$ is removed from the data.

Now we slightly modify SymCorrLDA by replacing Step 3-c in its generative process by:

3-c.    If ($x_i^{(\ell)} = m \neq \ell$), sample a topic $y_i^{(\ell)} \sim Multinomial(\boldsymbol{\theta}_d^{(m)})$

Figure 2(c) shows a graphical model representation of this alternative SymCorrLDA. In this model, non-pivot topics are dependent on the distribution behind the pivot topics, not dependent directly on the pivot topics as in the original SymCorrLDA. By this modification, the generative process is more naturally described. Accordingly, Eq. (2) of the full conditional probability is replaced by:

$$P(y_i^{(\ell)} = t, x_i^{(\ell)} = m | \mathbf{w}_i^{(\ell)} = w^{(\ell)}, \mathbf{y}_{-i}^{(\ell)}, \mathbf{z}^{(m)}, \mathbf{w}_{-i}^{(\ell)}, \mathbf{x}_{-i}, \beta^{(\ell)}, \gamma) \propto$$

$$\frac{n_{dm,-i} + \gamma}{n_{dm,-i} + \sum_{j \neq m} n_{dj} + L\gamma} \cdot \frac{C_{td}^{TD^{(m)}} + \alpha^{(m)}}{\sum_{t'} C_{t'd}^{TD^{(m)}} + T\alpha^{(m)}} \cdot \frac{C_{w^{(\ell)'} t,-i}^{W^{(\ell)}T} + \beta^{(\ell)}}{\sum_{w^{(\ell)'}} C_{w^{(\ell)'} t,-i}^{W^{(\ell)}T} + W^{(\ell)}\beta^{(\ell)}} \tag{3}$$

As you can see in the second term of the right-hand side above, the constraints are relaxed by this modification so that topics do not always have to be selected from the topics selected for the words with the pivot language flags, differently from that of Eq. (2). We will show through experiments how the modification affects the quality of the estimated multilingual topics, in the following section.

## 4   Experiments

In this section, we demonstrate some examples with SymCorrLDA, and then we compare multi-lingual topic models using various evaluation methods. For the evaluation, we use held-out log-likelihood using two datasets, the task of finding an English article that is on the same topic as that of a Japanese article, and a task with the languages reversed.

### 4.1   Settings

The datasets used in this work are two collections of Wikipedia articles: one is in English and Japanese, the other is in English, Japanese, and Spanish, and articles in each collection are connected across languages via inter-language links, as of November 2, 2009. We extracted text content from the original Wikipedia articles, removing link information and revision history information. We used WP2TXT[3] for this purpose. For English articles, we removed 418 types of standard stop words [12]. For Spanish articles, we removed 351 types of standard stop words [13]. As for Japanese articles, we removed function words, such as symbols, conjunctions and particles, using part-of-speech tags annotated by MeCab[4]. The statistics of the datasets after preprocessing are shown in Tables 1 and 2. We assumed each set of Wikipedia articles connected via inter-language links between two (or

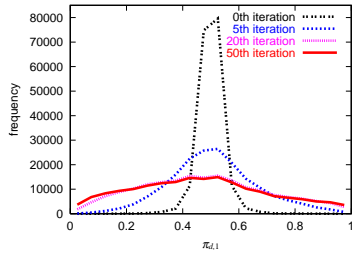

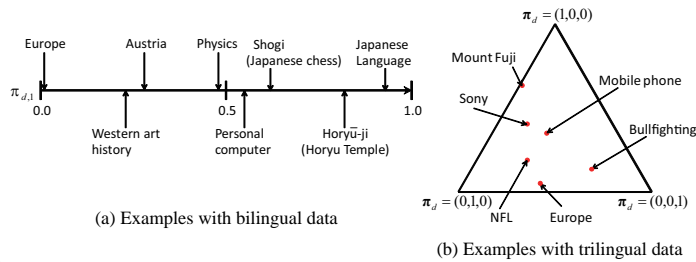

(a) Examples with bilingual data

(b) Examples with trilingual data

Figure 3: Change of frequency distribution of $\pi_{d,1}$ according to number of iterations

Figure 4: Document titles and corresponding $\pi_d$

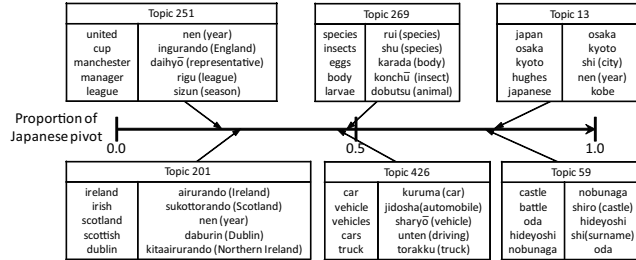

Figure 5: Topic examples and corresponding proportion of pivots assigned to Japanese. An English translation for each Japanese word follows in parentheses, except for Japanese proper nouns.

three) languages as a comparable document that consists of two (or three) language parts. To carry out the evaluation in the task of finding counterpart articles that we will describe later, we randomly divided the Wikipedia document collection at the document level into 80% *training documents* and 20% *test documents*. Furthermore, to compute held-out log-likelihood, we randomly divided each of the training documents at the word level into 90% *training set* and 10% *held-out set*.

We first estimated CI-LDA, SwitchLDA, CorrLDA, and SymCorrLDA and its alternative version ('SymCorrLDA-alt') as well as LDA for a baseline, using collapsed Gibbs sampling with the training set. In addition, we estimated a special implementation of SymCorrLDA, setting $\pi_d$ in a simple way for comparison, where the pivot language flag for each word is randomly selected according to the proportion of the length of each language part ('SymCorrLDA-rand').

For all the models, we assumed symmetric Dirichlet hyperparameters $\alpha = 50/T$ and $\beta = 0.01$, which have often been used in prior work [14]. We imposed the convergence condition of collapsed Gibbs sampling, such that the percentage change of held-out log-likelihood is less than 0.1%. For SymCorrLDA, we assumed symmetric Dirichlet hyperparameters $\gamma = 1$. For SwitchLDA, we assumed symmetric Dirichlet hyperparameters $\eta = 1$. We investigated the effect of $\gamma$ in SymCorrLDA and $\eta$ in SwitchLDA; however, the held-out log-likelihood was almost constant when varying these hyperparameters. LDA does not distinguish languages, so for a baseline we assumed all the language parts connected via inter-language links to be mixed together as a single document.

## 4.2 Pivot assignments

Figure 3 demonstrates how the frequency distribution of the pivot language-flag (binomial) parameter $\pi_{d,1}$ for the Japanese language with the bilingual dataset[5] in SymCorrLDA changes while in iterations of collapsed Gibbs sampling. This figure shows that the pivot language flag is randomly assigned at the initial state, and then it converges to an appropriate bias for each document as the iterations proceed. We next demonstrate how the pivot language flags are assigned to each document. Figure 4(a) shows the titles of eight documents and the corresponding $\pi_d$ when using the bilingual data ($T = 500$). If $\pi_{d,1}$ is close to 1, the article can be considered to be more related to a subject on Japanese or Japan. In contrast, if $\pi_{d,1}$ is close to 0 and therefore $\pi_{d,2} = 1 - \pi_{d,1}$ is close to 1, the article can be considered to be more related to a subject on English or English-speaking countries. Therefore, a pivot is assigned considering the language biases of the articles. Figure 4(b) shows the titles of six documents and the corresponding $\pi_d = (\pi_{d,1}, \pi_{d,2}, \pi_{d,3})$ when using the trilingual

Table 3: Per-word held-out log-likelihood with bilingual data. Boldface indicates the best result in each column.

|  | T=500 | | T=1000 | |
|---|---|---|---|---|
|  | Japanese | English | Japanese | English |
| LDA | -8.127 | -8.633 | -7.992 | -8.530 |
| CI-LDA | -8.136 | -8.644 | -8.008 | -8.549 |
| SwitchLDA | -8.139 | -8.641 | -8.012 | -8.549 |
| CorrLDA1 | -7.463 | -8.403 | -7.345 | -8.346 |
| CorrLDA2 | -7.777 | -8.197 | -7.663 | -8.109 |
| SymCorrLDA | **-7.433** | **-8.175** | **-7.317** | **-8.084** |
| SymCorrLDA-alt | -7.476 | -8.206 | -7.358 | -8.116 |
| SymCorrLDA-rand | -7.483 | -8.222 | -7.373 | -8.137 |

Table 4: Per-word held-out log-likelihood with trilingual data. Boldface indicates the best result in each column.

|  | T=500 | | | T=1000 | | |
|---|---|---|---|---|---|---|
|  | Japanese | English | Spanish | Japanese | English | Spanish |
| CorrLDA1 | -7.408 | -8.512 | -8.667 | -7.305 | -8.393 | -8.545 |
| CorrLDA2 | -7.655 | -8.198 | -8.467 | -7.572 | -8.122 | -8.401 |
| CorrLDA3 | -7.794 | -8.460 | -8.338 | -7.700 | -8.383 | -8.274 |
| SymCorrLDA | **-7.394** | **-8.178** | **-8.289** | **-7.287** | **-8.093** | **-8.215** |
| SymCorrLDA-alt | -7.440 | -8.209 | -8.330 | -7.330 | -8.120 | -8.254 |

data ($T = 500$). Here, $\pi_{d,1}$, $\pi_{d,2}$, and $\pi_{d,3}$ respectively indicate the pivot language-flag (multinomial) parameters corresponding to Japanese, English, and Spanish parts in each document. We further demonstrate the proportions of pivot assignments at the topic level. Figure 5 shows the content of 6 topics through 10 words with the highest probability for each language and for each topic when using the bilingual data ($T = 500$), some of which are biased to Japanese (Topics 13 and 59) or English (Topics 201 and 251), while the others have almost no bias. It can be seen that the pivot bias to specific languages can be interpreted.

## 4.3 Held-out log-likelihood

By measuring the held-out log-likelihood, we can evaluate the quality of each topic model. The higher the held-out log-likelihood, the greater the predictive ability of the model. In this work, we estimated multilingual topic models with the training set and computed the log-likelihood of generating the held-out set that was mentioned in Section 4.1.

Table 3 shows the held-out log-likelihood of each multilingual topic model estimated with the bilingual dataset when $T = 500$ and 1000. Note that the held-out log-likelihood (i.e., the micro-average per-word log-likelihood of the 10% held-out set) is shown for each language in this table, while the model estimation was performed over the 90% training set in all the languages. Hereafter, *CorrLDA1* refers to the CorrLDA model that was estimated when Japanese was the pivot language. As described in Section 2.3, the CorrLDA model first generates topics for the pivot language part of a document, and for the other language parts of the document, the model then uses the topics that were already generated in the pivot language. *CorrLDA2* refers to the CorrLDA model when English was the pivot language. As the results in Table 3 show, the held-out log-likelihoods of CorrLDA1 and CorrLDA2 are much higher than those of the other prior models: CI-LDA, SwitchLDA, and LDA, in both cases. This is because CorrLDA can capture direct dependencies between languages, due to the constraints that topics have to be selected from the topics selected in the pivot language parts. On the other hand, CI-LDA and SwitchLDA are too poorly constrained to effectively capture the dependencies between languages, as mentioned in Sections 2.1 and 2.2. In particular, CorrLDA1 has the highest held-out log-likelihood among all the prior models for Japanese, while CorrLDA2 is the best among all the prior models for English. This is probably due to the fact that CorrLDA can estimate topics from the pivot language parts (Japanese in the case of CorrLDA1) without any specific constraints; however, great constraints (topics having to be selected from the topics selected in the pivot language parts) are imposed for the other language parts. In SymCorrLDA, the held-out log-likelihood for Japanese is larger than that of CorrLDA1 (and the other models), and the held-out log-likelihood for English is larger than that of CorrLDA2. This is probably because SymCorrLDA estimates the pivot language appropriately adjusted for each word in each document. Next, we compare SymCorrLDA and its alternative version (SymCorrLDA-alt). We observed in Table 3 that the held-out log-likelihood of SymCorrLDA-alt is smaller than that of the original SymCorrLDA, and comparable to CorrLDA's best. This is because the constraints in SymCorrLDA-alt are relaxed so that topics do not always have to be selected from the topics selected for the words with the pivot language flags.

For further consideration, let us examine the results of the simplified implementation: SymCorrLDA-rand, which we defined in Section 4.1. SymCorrLDA-rand's held-out log-likelihood lies even below CorrLDA's best. These results reflect the fact that the performance of SymCorrLDA in its full form is inherently affected by the nature of the language biases in the multilingual comparable documents, rather than merely being affected by the language part length.

Table 4 shows the held-out log-likelihood with the trilingual data when $T = 500$ and 1000. Here, *CorrLDA3* refers to the CorrLDA model that was estimated when Spanish was the pivot language. As you can see in this table, SymCorrLDA's held-out log-likelihood is larger than CorrLDA's best. SymCorrLDA can estimate the pivot language appropriately adjusted for each word in each document in the trilingual data, as with the bilingual data. SymCorrLDA-alt behaves similarly as with the bilingual data.

For both the bilingual and trilingual data, the improvements with SymCorrLDA were statistically significant, compared to each of the other models, according to the Wilcoxon signed-rank test at the 5% level in terms of the word-by-word held-out log-likelihood. As for the scalability, SymCorrLDA is as scalable as CorrLDA because the time complexity of SymCorrLDA is the same order as that of CorrLDA: the number of topics times the sum of vocabulary size in each language. On clock time, SymCorrLDA does pay some extra, such as around 40% of the time for CorrLDA in the case of the bilingual data, for allocating the pivot language flags.

## 4.4 Finding counterpart articles

Given an article, we can find its unseen counterpart articles in other languages using a multilingual topic model. To evaluate this task, we experimented with the bilingual dataset. We estimated document-topic distributions of test documents for each language, using the topic-word distributions that were estimated by each multilingual topic model with training documents. We then evaluated the performance of finding English counterpart articles using Japanese articles as queries, and vice versa. For estimating the document-topic distributions of test documents, we used re-sampling of LDA using the topic-word distribution estimated beforehand by each multilingual topic model [15]. We then computed the Jensen-Shannon (JS) divergence [16] between a document-topic distribution of Japanese and that of English for each test document. Each held-out English-Japanese article pair connected via an inter-language link is considered to be on the same topic; therefore, JS divergence of such an article pair is expected to be small if the latent topic estimation is accurate. We first assumed each held-out Japanese article to be a query and the corresponding English article to be relevant, and evaluated the ranking of all the test articles of English in ascending order of the JS divergence; then we conducted the task with the languages reversed.

Table 5 shows the results of mean reciprocal rank (MRR), when $T = 500$ and 1000. The reciprocal rank is defined as the multiplicative inverse of the rank of the counterpart article corresponding to each query article, and the mean reciprocal rank is the average of it over all the query articles. CorrLDA works much more effectively than the other prior models: CI-LDA, SwitchLDA, and LDA, and overall, SymCor-

Table 5: MRR in counterpart article finding task. Boldface indicates the best result in each column.

| | Japanese to English | | English to Japanese | |
|---|---|---|---|---|
| | T=500 | T=1000 | T=500 | T=1000 |
| LDA | 0.0743 | 0.1027 | 0.0870 | 0.1262 |
| CI-LDA | 0.1426 | 0.1464 | 0.1697 | 0.1818 |
| SwitchLDA | 0.1357 | 0.1347 | 0.1668 | 0.1653 |
| CorrLDA1 | 0.2987 | 0.3281 | 0.2863 | 0.3111 |
| CorrLDA2 | 0.2829 | 0.3063 | 0.3161 | 0.3464 |
| SymCorrLDA | **0.3256** | **0.3592** | **0.3348** | **0.3685** |

rLDA works the most effectively. We observed that the improvements with SymCorrLDA were statistically significant according to the Wilcoxon signed-rank test at the 5% level, compared with each of the other models. Therefore, it is clear that SymCorrLDA estimates multilingual topics the most successfully in this experiment.

## 5 Conclusions

In this paper, we compared the performance of various topic models that can be applied to multilingual documents, not using multilingual dictionaries, in terms of held-out log-likelihood and in the task of cross-lingual link detection. We demonstrated through experiments that CorrLDA works significantly more effectively than CI-LDA, which was used in prior work on multilingual topic models. Furthermore, we proposed a new topic model, SymCorrLDA, that incorporates a hidden variable to control a pivot language, in an extension of CorrLDA. SymCorrLDA has an advantage in that it does not require a pivot language to be specified in advance, while CorrLDA does. We demonstrated that SymCorrLDA is more effective than CorrLDA and the other topic models, through experiments with Wikipedia datasets using held-out log-likelihood and in the task of finding counterpart articles in other languages. SymCorrLDA can be applied to other kinds of data that have multiple classes of representations, such as annotated image data. We plan to investigate this in future work.

**Acknowledgments**   We thank Sinead Williamson, Manami Matsuura, and the anonymous reviewers for valuable discussions and comments. This work was supported in part by the Grant-in-Aid for Scientific Research (#23300039) from JSPS, Japan.

## Footnotes

[1]Note that the term 'pivot language' does not have exactly the same meaning as that commonly used in the machine translation community, where it means an intermediary language for translation between more than three languages.

[2] $M_d^{(m)}$ words may indeed differ in size at the step of generating each word in the generative process. However, this is not problematic for inference, such as by collapsed Gibbs sampling, where any topic is first randomly assigned to every word, and a more appropriate topic is then re-assigned to each word, based on the topics previously assigned to all $N_d^{(m)}$ words, not $M_d^{(m)}$ words, with the pivot language flags.

[3]http://wp2txt.rubyforge.org/

[4]http://mecab.sourceforge.net/

[5]The parameter for English was $\pi_{d,2} = 1 - \pi_{d,1}$ in this case.

## References

[1] Thomas Hofmann. Probabilistic latent semantic indexing. In *Proceedings of the 22nd Anuual International ACM SIGIR Conference on Research and Development in Information Retrieval*, pages 50–57, Berkeley, California, USA, 1999.

[2] David M. Blei, Andrew Y. Ng, and Michael I. Jordan. Latent Dirichlet allocation. *Journal of Machine Learning Research*, 3:993–1022, 2003.

[3] David Mimno, Hanna M. Wallach, Jason Naradowsky, David A. Smith, and Andrew McCallum. Polylingual topic models. In *Proceedings of the 2009 Conference on Empirical Methods in Natural Language Processing*, pages 880–889, Stroudsburg, Pennsylvania, USA, 2009.

[4] Xiaochuan Ni, Jian-Tao Sun, Jian Hu, and Zheng Chen. Mining multilingual topics from wikipedia. In *Proceedings of the 18th International Conference on World Wide Web*, pages 1155–1156, Madrid, Spain, 2009.

[5] Elena Erosheva, Stephen Fienberg, and John Lafferty. Mixed-membership models of scientific publications. *Proceedings of the National Academy of Sciences of the United States of America*, 101:5220–5227, 2004.

[6] David Newman, Chaitanya Chemudugunta, Padhraic Smyth, and Mark Steyvers. Statistical entity-topic models. In *Proceedings of the 12th ACM SIGKDD International Conference on Knowledge Discovery and Data Mining*, pages 680–686, Philadelphia, Pennsylvania, USA, 2006.

[7] David M. Blei and Michael I. Jordan. Modeling annotated data. In *Proceedings of the 26th Annual International ACM SIGIR Conference on Research and Development in Informaion Retrieval*, pages 127–134, Toronto, Canada, 2003.

[8] Bing Zhao and Eric P. Xing. BiTAM: Bilingual topic admixture models for word alignment. In *Proceedings of the 44th Annual Meeting of the Association for Computational Linguistics*, pages 969–976, Sydney, Australia, 2006.

[9] Jordan Boyd-Graber and David M. Blei. Multilingual topic models for unaligned text. In *Proceedings of the 25th Conference on Uncertainty in Artificial Intelligence*, pages 75–82, Montreal, Canada, 2009.

[10] Jagadeesh Jagarlamudi and Hal Daume. Extracting multilingual topics from unaligned comparable corpora. In *Advances in Information Retrieval*, volume 5993 of *Lecture Notes in Computer Science*, pages 1–12. Springer, 2010.

[11] Duo Zhang, Qiaozhu Mei, and ChengXiang Zhai. Cross-lingual latent topic extraction. In *Proceedings of the 48th Annual Meeting of the Association for Computational Linguistics*, pages 1128–1137, Uppsala, Sweden, 2010.

[12] James P. Callan, W. Bruce Croft, and Stephen M. Harding. The INQUERY retrieval system. In *Proceedings of the 3rd International Conference on Database and Expert Systems Applications*, pages 78–83, Valencia, Spain, 1992.

[13] Jacques Savoy. Report on CLEF-2002 experiments: Combining multiple sources of evidence. In *Advances in Cross-Language Information Retrieval*, volume 2785 of *Lecture Notes in Computer Science*, pages 66–90. Springer, 2003.

[14] Mark Steyvers and Tom Griffiths. *Handbook of Latent Semantic Analysis*, chapter 21: Probabilistic Topic Models. Lawrence Erbaum Associates, Mahwah, New Jersey and London, 2007.

[15] Hanna M. Wallach, Iain Murray, Ruslan Salakhutdinov, and David Mimno. Evaluation methods for topic models. In *Proceedings of the 26th International Conference on Machine Learning*, pages 1105–1112, Montreal, Canada, 2009.

[16] Jianhua Lin. Divergence measures based on the shannon entropy. *IEEE Transactions on Information Theory*, 37(1):145–151, 1991.

